# Glove-TalkII: Mapping Hand Gestures to Speech Using Neural Networks

**S. Sidney Fels**
Department of Computer Science
University of Toronto
Toronto, ON, M5S 1A4
ssfels@ai.toronto.edu

**Geoffrey Hinton**
Department of Computer Science
University of Toronto
Toronto, ON, M5S 1A4
hinton@ai.toronto.edu

## Abstract

Glove-TalkII is a system which translates hand gestures to speech through an adaptive interface. Hand gestures are mapped continuously to 10 control parameters of a parallel formant speech synthesizer. The mapping allows the hand to act as an artificial vocal tract that produces speech in real time. This gives an unlimited vocabulary in addition to direct control of fundamental frequency and volume. Currently, the best version of Glove-TalkII uses several input devices (including a CyberGlove, a ContactGlove, a 3-space tracker, and a foot-pedal), a parallel formant speech synthesizer and 3 neural networks. The gesture-to-speech task is divided into vowel and consonant production by using a gating network to weight the outputs of a vowel and a consonant neural network. The gating network and the consonant network are trained with examples from the user. The vowel network implements a fixed, user-defined relationship between hand-position and vowel sound and does not require any training examples from the user. Volume, fundamental frequency and stop consonants are produced with a fixed mapping from the input devices. One subject has trained to speak intelligibly with Glove-TalkII. He speaks slowly with speech quality similar to a text-to-speech synthesizer but with far more natural-sounding pitch variations.

## 1  Introduction

There are many different possible schemes for converting hand gestures to speech. The choice of scheme depends on the granularity of the speech that you want to produce. Figure 1 identifies a spectrum defined by possible divisions of speech based on the duration of the sound for each granularity. What is interesting is that in general, the coarser the division of speech, the smaller the bandwidth necessary for the user. In contrast, where the granularity of speech is on the order of articulatory muscle movements (i.e. the artificial vocal tract [AVT]) high bandwidth control is necessary for good speech. Devices which implement this model of speech production are like musical instruments which produce speech sounds. The user must control the timing of sounds to produce speech much as a musician plays notes to produce music. The AVT allows unlimited vocabulary, control of pitch and non-verbal sounds. Glove-TalkII is an adaptive interface that implements an AVT.

Translating gestures to speech using an AVT model has a long history beginning in the late 1700's. Systems developed include a bellows-driven hand-varied resonator tube with auxiliary controls (1790's [9]), a rubber-moulded skull with actuators for manipulating tongue and jaw position (1880's [1]) and a keyboard-footpedal interface controlling a set of linearly spaced bandpass frequency generators called the Voder (1940 [3]). The Voder was demonstrated at the World's Fair in 1939 by operators who had trained continuously for one year to learn to speak with the system. This suggests that the task of speaking with a gestural interface is very difficult and the training times could be significantly decreased with a better interface. Glove-TalkII is implemented with neural networks which allows the system to learn the user's interpretation of an articulatory model of speaking.

This paper begins with an overview of the whole Glove-TalkII system. Then, each neural network is described along with its training and test results. Finally, a qualitative analysis is provided of the speech produced by a single subject after 100 hours of speaking with Glove-TalkII.

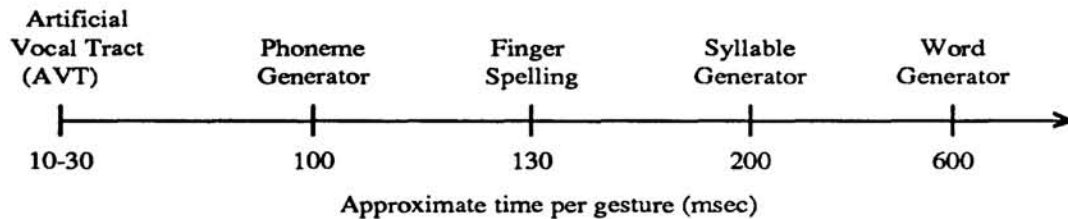

Figure 1: Spectrum of gesture-to-speech mappings based on the granularity of speech.

## 2  Overview of Glove-TalkII

The Glove-TalkII system converts hand gestures to speech, based on a gesture-to-formant model. The gesture vocabulary is based on a vocal-articulator model of the hand. By dividing the mapping tasks into independent subtasks, a substantial reduction in network size and training time is possible (see [4]).

Figure 2 illustrates the whole Glove-TalkII system. Important features include the

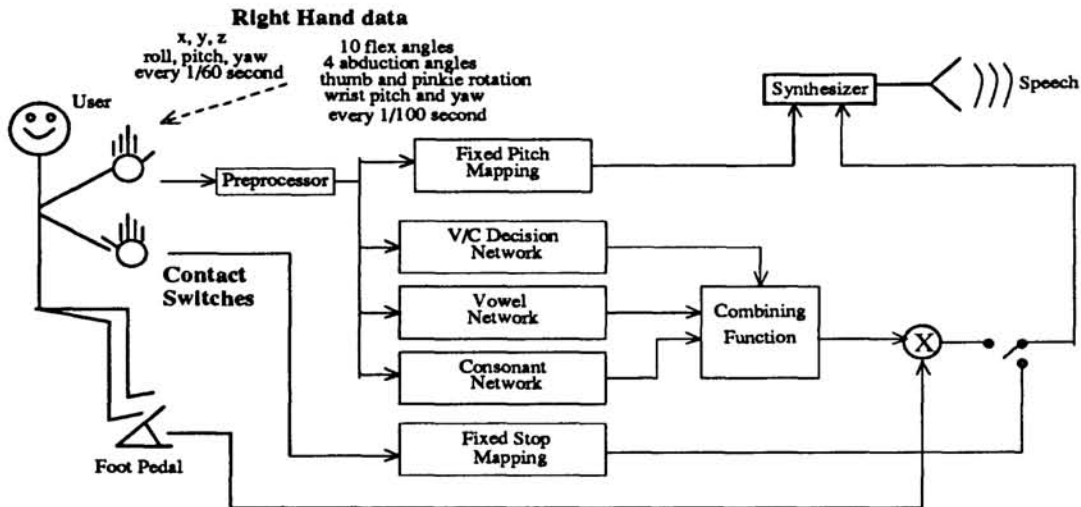

Figure 2: Block diagram of Glove-TalkII: input from the user is measured by the Cyberglove, polhemus, keyboard and foot pedal, then mapped using neural networks and fixed functions to formant parameters which drive the parallel formant synthesizer [8].

three neural networks labeled vowel/consonant decision (V/C), vowel, and consonant. The V/C network is trained on data collected from the user to decide whether he wants to produce a vowel or a consonant sound. Likewise, the consonant network is trained to produce consonant sounds based on user-generated examples based on an initial gesture vocabulary. In contrast, the vowel network implements a fixed mapping between hand-positions and vowel phonemes defined by the user. Nine contact points measured on the user's left hand by a ContactGlove designate the nine stop consonants (B, D, G, J, P, T, K, CH, NG), because the dynamics of such sounds proved too fast to be controlled by the user. The foot pedal provides a volume control by adjusting the speech amplitude and this mapping is fixed. The fundamental frequency, which is related to the pitch of the speech, is determined by a fixed mapping from the user's hand height. The output of the system drives 10 control parameters of a parallel formant speech synthesizer every 10 msec. The 10 control parameters are: nasal amplitude (ALF), first, second and third formant frequency and amplitude (F1, A1, F2, A2, F3, A3), high frequency amplitude (AHF), degree of voicing (V) and fundamental frequency (F0). Each of the control parameters is quantized to 6 bits.

Once trained, Glove-TalkII can be used as follows: to initiate speech, the user forms the hand shape of the first sound she intends to produce. She depresses the foot pedal and the sound comes out of the synthesizer. Vowels and consonants of various qualities are produced in a continuous fashion through the appropriate co-ordination of hand and foot motions. Words are formed by making the correct motions; for example, to say "hello" the user forms the "h" sound, depresses the foot pedal and quickly moves her hand to produce the "e" sound, then the "l" sound and finally the "o" sound. The user has complete control of the timing and quality of the individual sounds. The articulatory mapping between gestures and speech

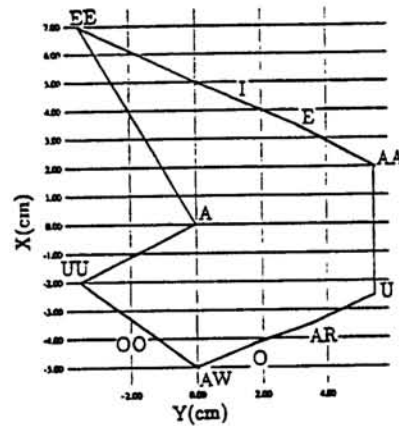

Figure 3: Hand-position to Vowel Sound Mapping. The coordinates are specified relative to the origin at the sound A. The X and Y coordinates form a horizontal plane parallel to the floor when the user is sitting. The 11 cardinal phoneme targets are determined with the text-to-speech synthesizer.

is decided *a priori*. The mapping is based on a simplistic articulatory phonetic description of speech [5]. The X,Y coordinates (measured by the polhemus) are mapped to something like tongue position and height[1] producing vowels when the user's hand is in an open configuration (see figure 2 for the correspondence and table 1 for a typical vowel configuration). Manner and place of articulation for non-stop consonants are determined by opposition of the thumb with the index and middle fingers as described in table 1. The ring finger controls voicing. Only *static* articulatory configurations are used as training points for the neural networks, and the interpolation between them is a result of the learning but is not explicitly trained. Ideally, the transitions should also be learned, but in the text-to-speech formant data we use for training [6] these transitions are poor, and it is very hard to extract formant trajectories from real speech accurately.

## 2.1   The Vowel/Consonant (V/C) Network

The V/C network decides, on the basis of the current configuration of the user's hand, to emit a vowel or a consonant sound. For the quantitative results reported here, we used a 10-5-1 feed-forward network with sigmoid activations [7]. The 10 inputs are ten scaled hand parameters measured with a Cyberglove: 8 flex angles (knuckle and middle joints of the thumb, index, middle and ring fingers), thumb abduction angle and thumb rotation angle. The output is a single number representing the probability that the hand configuration indicates a vowel. The output of the V/C network is used to gate the outputs of the vowel and consonant networks, which then produce a mixture of vowel and consonant formant parameters. The training data available includes only user-produced vowel or consonant sounds. The network interpolates between hand configurations to create a smooth but fairly rapid transition between vowels and consonants.

For quantitative analysis, typical training data consists of 2600 examples of consonant configurations (350 approximants, 1510 fricatives [and aspirant], and 740 nasals) and 700 examples of vowel configurations. The consonant examples were obtained from training data collected for the consonant network by an expert user. The vowel examples were collected from the user by requiring him to move his hand in vowel configurations for a specified amount of time. This procedure was performed in several sessions. The test set consists of 1614 examples (1380 consonants and 234 vowels). After training,[2] the mean squared error on the training and test

| | | | | |
|---|---|---|---|---|
|  |  |  |  |  |
| DH | F | H | L | M |
|  |  |  |  |  |
| N | R | S | SH | TH |
|  |  |  |  |  |
| V | W | Z | ZH | vowel |

Table 1: Static Gesture-to-Consonant Mapping for all phonemes. Note, each gesture corresponds to a static *non-stop* consonant phoneme generated by the text-to-speech synthesizer.

set was less than $10^{-4}$.

During normal speaking neither network made perceptual errors. The decision boundary feels quite sharp, and provides very predictable, quick transitions from vowels to consonants and back. Also, vowel sounds are produced when the user hyperextends his hand. Any unusual configurations that would intuitively be expected to produce consonant sounds do indeed produce consonant sounds.

## 2.2 The Vowel Network

The vowel network is a 2-11-8 feed forward network. The 11 hidden units are normalized radial basis functions (RBFs) [2] which are centered to respond to one of 11 cardinal vowels. The outputs are sigmoid units representing 8 synthesizer control parameters (ALF, F1, A1, F2, A2, F3, A3, AHF). The radial basis function used is:

$$o_j = e^{-\frac{\sum (w_{ji} - o_i)^2}{\sigma_j^2}} \tag{1}$$

where $o_j$ is the (un-normalized) output of the RBF unit, $w_{ji}$ is the weight from unit $i$ to unit $j$, $o_i$ is the output of input unit $i$, and $\sigma_j^2$ is the variance of the RBF. The normalization used is:

$$n_j = \frac{o_j}{\sum_{m \in P} o_m} \tag{2}$$

where $n_j$ is the normalized output of unit $j$ and the summation is over all the units in the group of normalized RBF units. The centres of the RBF units are fixed

conjugate gradient descent and a line search.

according to the X and Y values of each of the 11 vowels in the predefined mapping (see figure 2). The variances of the 11 RBF's are set to 0.025.

The weights from the RBF units to the output units are trained. For the training data, 100 identical examples of each vowel are generated from their corresponding X and Y positions in the user-defined mapping, providing 1100 examples. Noise is then added to the *scaled* X and Y coordinates for each example. The added noise is uniformly distributed in the range -0.025 to 0.025. In terms of unscaled ranges, these correspond to an X range of approximately $\pm$ 0.5 cm and a Y range of $\pm$ 0.26 cm.

Three different test sets were created. Each test set had 50 examples of each vowel for a total of 550 examples. The first test set used additive uniform noise in the interval $\pm$ 0.025. The second and third test sets used additive uniform noise in the interval $\pm$ 0.05 and $\pm$ 0.1 respectively.

The mean squared error on the training set was 0.0016. The MSE on the additive noise test sets (noise = $\pm$ 0.025, 0.05 and 0.01) was 0.0018, 0.0038, 0.0120 which corresponds to expected errors of 1.1%, 3.1% and 5.5% in the formant parameters, respectively. This network performs well perceptually. The key feature is the normalization of the RBF units. Often, when speaking, the user will overshoot cardinal vowel positions (especially when she is producing dipthongs) and all the RBF units will be quite suppressed. However, the normalization magnifies any slight difference between the activities of the units and the sound produced will be dominated by the cardinal vowel corresponding to the one whose centre is closest in hand space.

## 2.3   The Consonant Network

The consonant network is a 10-14-9 feed-forward network. The 14 hidden units are normalized RBF units. Each RBF is centred at a hand configuration determined from training data collected from the user corresponding to one of 14 static consonant phonemes. The target consonants are created with a text-to-speech synthesizer. Figure 1 defines the initial mapping for each of the 14 consonants. The 9 sigmoid output units represent 9 control parameters of the formant synthesizer (ALF, F1, A1, F2, A2, F3, A3, AHF, V). The voicing parameter is required since consonant sounds have different degrees of voicing. The inputs are the same as for the manager V/C network.

Training and test data for the consonant network is obtained from the user. Target data is created for each of the 14 consonant sounds using the text-to-speech synthesizer. The scheme to collect data for a single consonant is:

1. The target consonant is played for 100 msec through the speech synthesizer;

2. the user forms a hand configuration corresponding to the consonant;

3. the user depresses the foot pedal to begin recording; the start of recording is indicated by the appearance of a green square;

4. 10-15 time steps of hand data are collected and stored with the corresponding formant parameter targets and phoneme identifier; the end of data collection is indicated by turning the green square red;

5. the user chooses whether to save the data to a file, and whether to redo the current target or move to the next one.

Using this procedure 350 approximants, 1510 fricatives and 700 nasals were collected and scaled for the training data. The hand data were averaged for each consonant sound to form the RBF centres. For the test data, 255 approximants, 960 fricatives and 165 nasals were collected and scaled. The RBF variances were set to 0.05.

The mean square error on the training set was 0.005 and on the testing set was 0.01 corresponding to expected errors of 3.3% and 4.7% in the formant parameters, respectively. Listening to the output of the network reveals that each sound is produced reasonably well when the user's hand is held in a fixed position. The only difficulty is that the R and L sounds are very sensitive to motion of the index finger.

## 3   Qualitative Performance of Glove-TalkII

One subject, who is an accomplished pianist, has been trained extensively to speak with Glove-TalkII. We expected that his pianistic skill in forming finger patterns and his musical training would help him learn to speak with Glove-TalkII. After 100 hours of training, his speech with Glove-TalkII is intelligible and somewhat natural-sounding. He still finds it difficult to speak quickly, pronounce polysyllabic words, and speak spontaneously.

During his training, Glove-TalkII also adapted to suit changes required by the subject. Initially, good performance of the V/C network is critical for the user to learn to speak. If the V/C network performs poorly the user hears a mixture of vowel and consonant sounds making it difficult to adjust his hand configurations to say different utterances. For this reason, it is important to have the user comfortable with the initial mapping so that the training data collected leads to the V/C network performing well. In the 100 hours of practice, Glove-TalkII was retrained about 10 times. Four significant changes were made from the original system analysed here for the new subject. First, the NG sound was added to the non-stop consonant list by adding an additional hand shape, namely the user touches his pinkie to his thumb on his right hand. To accomodate this change, the consonant and V/C network had two inputs added to represent the two flex angles of the pinkie. Also, the consonant network has an extra hidden unit for the NG sound. Second, the consonant network was trained to allow the RBF centres to change. After the hidden-to-output weights were trained until little improvement was seen, the input-to-hidden weights (i.e. the RBF centres) were also allowed to adapt. This noticeably improved performance for the user. Third, the vowel mapping was altered so that the I was moved closer to the EE sound and the entire mapping was reduced to 75% of its size. Fourth, for this subject, the V/C network needed was a 10-10-1 feed-forward sigmoid unit network. Understanding the interaction between the user's adaptation and Glove-TalkII's adaptation remains an interesting research pursuit.

## 4   Summary

The initial mapping is loosely based on an articulatory model of speech. An open configuration of the hand corresponds to an unobstructed vocal tract, which in turn generates vowel sounds. Different vowel sounds are produced by movements of the hand in a horizontal X-Y plane that corresponds to movements of the first two formants which are roughly related to tongue position. Consonants other than stops are produced by closing the index, middle, or ring fingers or flexing the thumb, representing constrictions in the vocal tract. Stop consonants are produced by

contact switches worn on the user's left hand. F0 is controlled by hand height and speaking intensity by foot pedal depression.

Glove-TalkII learns the user's interpretation of this initial mapping. The V/C network and the consonant network learn the mapping from examples generated by the user during phases of training. The vowel network is trained on examples computed from the user-defined mapping between hand-position and vowels. The F0 and volume mappings are non-adaptive.

One subject was trained to use Glove-TalkII. After 100 hours of practice he is able to speak intelligibly. His speech is fairly slow (1.5 to 3 times slower than normal speech) and somewhat robotic. It sounds similar to speech produced with a text-to-speech synthesizer but has a more natural intonation contour which greatly improves the intelligibility and naturalness of the speech. Reading novel passages intelligibly usually requires several attempts, especially with polysyllabic words. Intelligible spontaneous speech is possible but difficult.

## Acknowledgements

We thank Peter Dayan, Sageev Oore and Mike Revow for their contributions. This research was funded by the Institute for Robotics and Intelligent Systems and NSERC. Geoffrey Hinton is the Noranda fellow of the Canadian Institute for Advanced Research.

## Footnotes

[1]In reality, the XY coordinates map more closely to changes in the first two formants, F1 and F2 of vowels. From the user's perspective though, the link to tongue movement is useful.

[2]The V/C network, the vowel network and the consonant network are trained using

## References

[1] A. G. Bell. Making a talking-machine. In *Beinn Bhreagh Recorder*, pages 61–72, November 1909.

[2] D. Broomhead and D. Lowe. Multivariable functional interpolation and adaptive networks. *Complex Systems*, 2:321–355, 1988.

[3] Homer Dudley, R. R. Riesz, and S. S. A. Watkins. A synthetic speaker. *Journal of the Franklin Institute*, 227(6):739–764, June 1939.

[4] S. S. Fels. Building adaptive interfaces using neural networks: The Glove-Talk pilot study. Technical Report CRG-TR-90-1, University of Toronto, 1990.

[5] P. Ladefoged. *A course in Phonetics (2 ed.)*. Harcourt Brace Javanovich, New York, 1982.

[6] E. Lewis. A 'C' implementation of the JSRU text-to-speech system. Technical report, Computer Science Dept., University of Bristol, 1989.

[7] D. E. Rumelhart, G. E. Hinton, and R. J. Williams. Learning internal representations by back-propagating errors. *Nature*, *323*:533–536, 1986.

[8] J. M. Rye and J. N. Holmes. A versatile software parallel-formant speech synthesizer. Technical Report JSRU-RR-1016, Joint Speech Research Unit, Malvern, UK, 1982.

[9] Wolfgang Ritter von Kempelen. *Mechanismus der menschlichen Sprache nebst Beschreibungeiner sprechenden Maschine. Mit einer Einleitung vonHerbert E. Brekle und Wolfgang Wild.* Stuttgart-Bad Cannstatt F. Frommann, Stuttgart, 1970.